# The Relaxed Online Maximum Margin Algorithm

**Yi Li and Philip M. Long**
Department of Computer Science
National University of Singapore
Singapore 119260, Republic of Singapore
{*liyi,plong*}*@comp.nus.edu.sg*

## Abstract

We describe a new incremental algorithm for training linear threshold functions: the Relaxed Online Maximum Margin Algorithm, or ROMMA. ROMMA can be viewed as an approximation to the algorithm that repeatedly chooses the hyperplane that classifies previously seen examples correctly with the maximum margin. It is known that such a maximum-margin hypothesis can be computed by minimizing the length of the weight vector subject to a number of linear constraints. ROMMA works by maintaining a relatively simple relaxation of these constraints that can be efficiently updated. We prove a mistake bound for ROMMA that is the same as that proved for the perceptron algorithm. Our analysis implies that the more computationally intensive maximum-margin algorithm also satisfies this mistake bound; this is the first worst-case performance guarantee for this algorithm. We describe some experiments using ROMMA and a variant that updates its hypothesis more aggressively as batch algorithms to recognize handwritten digits. The computational complexity and simplicity of these algorithms is similar to that of perceptron algorithm, but their generalization is much better. We describe a sense in which the performance of ROMMA converges to that of SVM in the limit if bias isn't considered.

## 1  Introduction

The perceptron algorithm [10, 11] is well-known for its simplicity and effectiveness in the case of linearly separable data. Vapnik's support vector machines (SVM) [13] use quadratic programming to find the weight vector that classifies all the training data correctly and maximizes the margin, i.e. the minimal distance between the separating hyperplane and the instances. This algorithm is slower than the perceptron algorithm, but generalizes better. On the other hand, as an incremental algorithm, the perceptron algorithm is better suited for online learning, where the algorithm repeatedly must classify patterns one at a time, then finds out the correct classification, and then updates its hypothesis before making the next prediction.

In this paper, we design and analyze a new simple online algorithm called ROMMA (the Relaxed Online Maximum Margin Algorithm) for classification using a linear threshold

function. ROMMA has similar time complexity to the perceptron algorithm, but its generalization performance in our experiments is much better on average. Moreover, ROMMA can be applied with kernel functions.

We conducted experiments similar to those performed by Cortes and Vapnik [2] and Freund and Schapire [3] on the problem of handwritten digit recognition. We tested the standard perceptron algorithm, the voted perceptron algorithm (for details, see [3]) and our new algorithm, using the polynomial kernel function with $d = 4$ (the choice that was best in [3]). We found that our new algorithm performed better than the standard perceptron algorithm, had slightly better performance than the voted perceptron.

For some other research with aims similar to ours, we refer the reader to [9, 4, 5, 6].

The paper is organized as follows. In Section 2, we describe ROMMA in enough detail to determine its predictions, and prove a mistake bound for it. In Section 3, we describe ROMMA in more detail. In Section 4, we compare the experimental results of ROMMA and an aggressive variant of ROMMA with the perceptron and the voted perceptron algorithms.

## 2 A mistake-bound analysis

### 2.1 The online algorithms

For concreteness, our analysis will concern the case in which instances (also called patterns) and weight vectors are in $\mathbf{R}^n$. Fix $n \in \mathbf{N}$. In the standard online learning model [7], learning proceeds in *trials*. In the $t$th trial, the algorithm is first presented with an instance $\vec{x}_t \in \mathbf{R}^n$. Next, the algorithm outputs a prediction $\hat{y}_t$ of the classification of $\vec{x}_t$. Finally, the algorithm finds out the correct classification $y_t \in \{-1, 1\}$. If $\hat{y}_t \neq y_t$, then we say that the algorithm makes a mistake. It is worth emphasizing that in this model, when making its prediction for the $t$th trial, the algorithm only has access to instance-classification pairs for previous trials.

All of the online algorithms that we will consider work by maintaining a weight vector $\vec{w}_t$ which is updated between trials, and predicting $\hat{y}_t = \text{sign}(\vec{w}_t \cdot \vec{x}_t)$, where $\text{sign}(z)$ is 1 if $z$ is positive, $-1$ if $z$ is negative, and 0 otherwise.[1]

**The perceptron algorithm.** The perceptron algorithm, due to Rosenblatt [10, 11], starts off with $\vec{w}_1 = 0$. When its prediction differs from the label $y_t$, it updates its weight vector by $\vec{w}_{t+1} = \vec{w}_t + y_t \vec{x}_t$. If the prediction is correct then the weight vector is not changed.

The next three algorithms that we will consider assume that all of the data seen by the online algorithm is collectively linearly separable, i.e. that there is a weight vector $\vec{u}$ such that for all each trial $t$, $y_t = \text{sign}(\vec{u} \cdot \vec{x}_t)$. When kernel functions are used, this is often the case in practice.

**The ideal online maximum margin algorithm.** On each trial $t$, this algorithm chooses a weight vector $\vec{w}_t$ for which for all previous trials $s \leq t$, $\text{sign}(\vec{w}_t \cdot \vec{x}_s) = y_s$, and which maximizes the minimum distance of any $\vec{x}_s$ to the separating hyperplane. It is known [1, 14] that this can be implemented by choosing $\vec{w}_t$ to minimize $\|\vec{w}_t\|$ subject to the constraints that $y_s(\vec{w}_t \cdot \vec{x}_s) \geq 1$ for all $s \leq t$. These constraints define a convex polyhedron in weight space which we will refer to as $P_t$.

**The relaxed online maximum margin algorithm.** This is our new algorithm. The first difference is that trials in which mistakes are not made are ignored. The second difference

is in how the algorithm responds to mistakes. The relaxed algorithm starts off like the ideal algorithm. Before the second trial, it sets $\vec{w}_2$ to be the shortest weight vector such that $y_1(\vec{w}_2 \cdot \vec{x}_1) \geq 1$. If there is a mistake on the second trial, it chooses $\vec{w}_3$ as would the ideal algorithm, to be the smallest element of

$$\{\vec{w} : y_1(\vec{w} \cdot \vec{x}_1) \geq 1\} \cap \{\vec{w} : y_2(\vec{w} \cdot \vec{x}_2) \geq 1\}. \tag{1}$$

However, if the third trial is a mistake, then it behaves differently. Instead of choosing $\vec{w}_4$ to be the smallest element of

$$\{\vec{w} : y_1(\vec{w} \cdot \vec{x}_1) \geq 1\} \cap \{\vec{w} : y_2(\vec{w} \cdot \vec{x}_2) \geq 1\} \cap \{\vec{w} : y_3(\vec{w} \cdot \vec{x}_3) \geq 1\},$$

it lets $\vec{w}_4$ be the smallest element of

$$\{\vec{w} : \vec{w}_3 \cdot \vec{w} \geq ||\vec{w}_3||^2\} \cap \{\vec{w} : y_3(\vec{w} \cdot \vec{x}_3) \geq 1\}.$$

This can be thought of as, before the third trial, replacing the polyhedron defined by (1) with the halfspace $\{\vec{w} : \vec{w}_3 \cdot \vec{w} \geq ||\vec{w}_3||^2\}$ (see Figure 1).

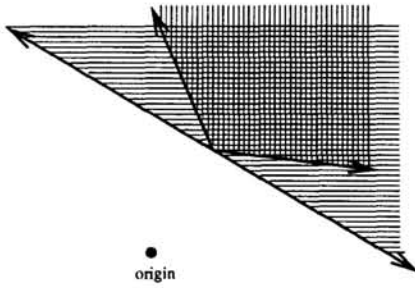

Figure 1: In ROMMA, a convex polyhedron in weight space is replaced with the halfspace with the same smallest element.

Note that this halfspace contains the polyhedron of (1); in fact, it contains any convex set whose smallest element is $\vec{w}_3$. Thus, it can be thought of as the least restrictive convex constraint for which the smallest satisfying weight vector is $\vec{w}_3$. Let us call this halfspace $H_3$. The algorithm continues in this manner. If the $t$th trial is a mistake, then $\vec{w}_{t+1}$ is chosen to be the smallest element of $H_t \cap \{\vec{w} : y_t(\vec{w} \cdot \vec{x}_t) \geq 1\}$, and $H_{t+1}$ is set to be $\{\vec{w} : \vec{w}_{t+1} \cdot \vec{w} \geq ||\vec{w}_{t+1}||^2\}$. If the $t$th trial is not a mistake, then $\vec{w}_{t+1} = \vec{w}_t$ and $H_{t+1} = H_t$. We will call $H_t$ the old constraint, and $\{\vec{w} : y_t(\vec{w} \cdot \vec{x}_t) \geq 1\}$ the new constraint.

Note that after each mistake, this algorithm needs only to solve a quadratic programming problem with two linear constraints. In fact, there is a simple closed-form expression for $\vec{w}_{t+1}$ as a function of $\vec{w}_t$, $\vec{x}_t$ and $y_t$ that enables it to be computed incrementally using time similar to that of the perceptron algorithm. This is described in Section 3.

**The relaxed online maximum margin algorithm with aggressive updating.** The algorithm is the same as the previous algorithm, except that an update is made after any trial in which $y_t(\vec{w}_t \cdot \vec{x}_t) < 1$, not just after mistakes.

## 2.2 Upper bound on the number of mistakes made

Now we prove a bound on the number of mistakes made by ROMMA. As in previous mistake bound proofs (e.g. [8]), we will show that mistakes result in an increase in a "measure of progress", and then appeal to a bound on the total possible progress. Our proof will use the squared length of $\vec{w}_t$ as its measure of progress.

First we will need the following lemmas.

**Lemma 1** *On any run of ROMMA on linearly separable data, if trial $t$ was a mistake, then the new constraint is binding at the new weight vector, i.e. $y_t(\vec{w}_{t+1} \cdot \vec{x}_t) = 1$.*

**Proof:** For the purpose of contradiction suppose the new constraint is not binding at the new weight vector $\vec{w}_{t+1}$. Since $\vec{w}_t$ fails to satisfy this constraint, the line connecting $\vec{w}_{t+1}$ and $\vec{w}_t$ intersects with the border hyperplane of the new constraint, and we denote the intersecting point $\vec{w}_q$. Then $\vec{w}_q$ can be represented as $\vec{w}_q = \alpha\vec{w}_t + (1-\alpha)\vec{w}_{t+1}, 0 < \alpha < 1$.

Since the square of Euclidean length $\|\cdot\|^2$ is a convex function, the following holds:

$$\|\vec{w}_q\|^2 \le \alpha\|\vec{w}_t\|^2 + (1-\alpha)\|\vec{w}_{t+1}\|^2$$

Since $\vec{w}_t$ is the unique smallest member of $H_t$ and $\vec{w}_{t+1} \ne \vec{w}_t$, we have $\|\vec{w}_t\|^2 < \|\vec{w}_{t+1}\|^2$, which implies

$$\|\vec{w}_q\|^2 < \|\vec{w}_{t+1}\|^2 \tag{2}$$

Since $\vec{w}_t$ and $\vec{w}_{t+1}$ are both in $H_t$, $\vec{w}_q$ is too, and hence (2) contradicts the definition of $\vec{w}_{t+1}$. $\quad\square$

**Lemma 2** *On any run of ROMMA on linearly separable data, if trial $t$ was a mistake, and not the first one, then the old constraint is binding at the new weight vector, i.e. $\vec{w}_{t+1} \cdot \vec{w}_t = \|\vec{w}_t\|^2$.*

**Proof:** Let $A_t$ be the plane of weight vectors that make the new constraint tight, i.e. $A_t = \{\vec{w} : y_t(\vec{w} \cdot \vec{x}_t) = 1\}$. By Lemma 1, $\vec{w}_{t+1} \in A_t$. Let $\vec{a}_t = y_t\vec{x}_t/\|\vec{x}_t\|^2$ be the element of $A_t$ that is perpendicular to it. Then each $\vec{w} \in A_t$ satisfies $\|\vec{w}\|^2 = \|\vec{a}_t\|^2 + \|\vec{w} - \vec{a}_t\|^2$, and therefore the length of a vector $\vec{w}$ in $A_t$ is minimized when $\vec{w} = \vec{a}_t$ and is monotone in the distance from $\vec{w}$ to $\vec{a}_t$. Thus, if the old constraint is not binding, then $\vec{w}_{t+1} = \vec{a}_t$, since otherwise the solution could be improved by moving $\vec{w}_{t+1}$ a little bit toward $\vec{a}_t$. But the old constraint requires that $(\vec{w}_t \cdot \vec{w}_{t+1}) \ge \|\vec{w}_t\|^2$, and if $\vec{w}_{t+1} = \vec{a}_t = y_t\vec{x}_t/\|\vec{x}_t\|^2$, this means that $\vec{w}_t \cdot (y_t\vec{x}_t/\|\vec{x}_t\|^2) \ge \|\vec{w}_t\|^2$. Rearranging, we get $y_t(\vec{w}_t \cdot \vec{x}_t) \ge \|\vec{x}_t\|^2\|\vec{w}_t\|^2 > 0$ ($\|x_t\| > 0$ follows from the fact that the data is linearly separable, and $\|w_t\| > 0$ follows from the fact that there was at least one previous mistake). But since trial $t$ was a mistake, $y_t(\vec{w}_t \cdot \vec{x}_t) \le 0$, a contradiction. $\quad\square$

Now we're ready to prove the mistake bound.

**Theorem 3** *Choose $m \in \mathbf{N}$, and a sequence $(\vec{x}_1, y_1), \cdots, (\vec{x}_m, y_m)$ of pattern-classification pairs in $\mathbf{R}^n \times \{-1, +1\}$. Let $R = \max_t \|\vec{x}_t\|$. If there is a weight vector $\vec{u}$ such that $y_t(\vec{u} \cdot \vec{x}_t) \ge 1$ for all $1 \le t \le m$, then the number of mistakes made by ROMMA on $(\vec{x}_1, y_1), \cdots, (\vec{x}_m, y_m)$ is at most $R^2\|\vec{u}\|^2$.*

**Proof:** First, we claim that for all $t$, $\vec{u} \in H_t$. This is easily seen since $\vec{u}$ satisfies all the constraints that are ever imposed on a weight vector, and therefore all relaxations of such constraints. Since $\vec{w}_t$ is the smallest element of $H_t$, we have $\|\vec{w}_t\| \le \|\vec{u}\|$.

We have $\vec{w}_2 = y_1\vec{x}_1/\|\vec{x}_1\|^2$, and therefore $\|\vec{w}_2\| = 1/\|\vec{x}_1\| \ge 1/R$ which implies $\|\vec{w}_2\|^2 \ge 1/R^2$. We claim that if any trial $t > 1$ is a mistake, then $\|\vec{w}_{t+1}\|^2 \ge \|\vec{w}_t\|^2 + 1/R^2$. This will imply by induction that after $M$ mistakes, the squared length of the algorithm's weight vector is at least $M/R^2$, which, since all of the algorithm's weight vectors are no longer than $\vec{u}$, will complete the proof.

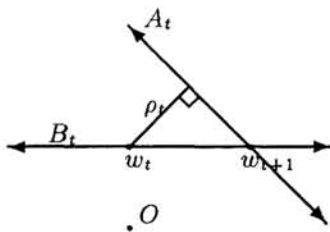

Figure 2: $A_t$, $B_t$, and $\rho_t$

Choose an index $t > 1$ of a trial in which a mistake is made. Let $A_t = \{\vec{w} : y_t(\vec{w} \cdot \vec{x}_t) = 1\}$ and $B_t = \{\vec{w} : (\vec{w} \cdot \vec{w}_t) = \|\vec{w}_t\|^2\}$. By Lemmas 1 and 2, $\vec{w}_{t+1} \in A_t \cap B_t$.

The distance from $\vec{w}_t$ to $A_t$ (call it $\rho_t$) satisfies

$$\rho_t = \frac{|y_t(\vec{x}_t \cdot \vec{w}_t) - 1|}{\|\vec{x}_t\|} \ge \frac{1}{\|\vec{x}_t\|} \ge \frac{1}{R}, \tag{3}$$

since the fact that there was a mistake in trial $t$ implies $y_t(\vec{x}_t \cdot \vec{w}_t) \le 0$. Also, since $\vec{w}_{t+1} \in A_t$,

$$\|w_{t+1} - w_t\| \ge \rho_t. \tag{4}$$

Because $\vec{w}_t$ is the normal vector of $B_t$ and $\vec{w}_{t+1} \in B_t$, we have

$$\|\vec{w}_{t+1}\|^2 = \|\vec{w}_t\|^2 + \|\vec{w}_{t+1} - \vec{w}_t\|^2.$$

Thus, applying (3) and (4), we have $\|\vec{w}_{t+1}\|^2 - \|\vec{w}_t\|^2 = \|\vec{w}_{t+1} - \vec{w}_t\|^2 \geq \rho_t^2 \geq 1/R^2$, which, as discussed above, completes the proof. □

Using the fact, easily proved using induction, that for all $t$, $P_t \subseteq H_t$, we can easily prove the following, which complements analyses of the maximum margin algorithm using independence assumptions [1, 14, 12]. Details are omitted due to space constraints.

**Theorem 4** *Choose $m \in \mathbf{N}$, and a sequence $(\vec{x}_1, y_1), \cdots, (\vec{x}_m, y_m)$ of pattern-classification pairs in $\mathbf{R}^n \times \{-1, +1\}$. Let $R = \max_t \|\vec{x}_t\|$. If there is a weight vector $\vec{u}$ such that $y_t(\vec{u} \cdot \vec{x}_t) \geq 1$ for all $1 \leq t \leq m$, then the number of mistakes made by the ideal online maximum margin algorithm on $(\vec{x}_1, y_1), \cdots, (\vec{x}_m, y_m)$ is at most $R^2\|\vec{u}\|^2$.*

In the proof of Theorem 3, if an update is made and $y_t(\vec{w}_t \cdot \vec{x}_t) < 1 - \delta$ instead of $y_t(\vec{w}_t \cdot \vec{x}_t) \leq 0$, then the progress made can be seen to be at least $\delta^2/R^2$. This can be applied to prove the following.

**Theorem 5** *Choose $\delta > 0$, $m \in \mathbf{N}$, and a sequence $(\vec{x}_1, y_1), \cdots, (\vec{x}_m, y_m)$ of pattern-classification pairs in $\mathbf{R}^n \times \{-1, +1\}$. Let $R = \max_t \|\vec{x}_t\|$. If there is a weight vector $\vec{u}$ such that $y_t(\vec{u} \cdot \vec{x}_t) \geq 1$ for all $1 \leq t \leq m$, then if $(\vec{x}_1, y_1), \cdots, (\vec{x}_m, y_m)$ are presented on line the number of trials in which aggressive ROMMA has $y_t(\vec{w}_t \cdot \vec{x}_t) < 1 - \delta$ is at most $R^2\|\vec{u}\|^2/\delta^2$.*

Theorem 5 implies that, in a sense, repeatedly cycling through a dataset using aggressive ROMMA will eventually converge to SVM; note however that bias is not considered.

## 3  An efficient implementation

When the prediction of ROMMA differs from the expected label, the algorithm chooses $\vec{w}_{t+1}$ to minimize $\|\vec{w}_{t+1}\|$ subject to $A\vec{w}_{t+1} = b$, where $A = \begin{pmatrix} \vec{w}_t^T \\ \vec{x}_t^T \end{pmatrix}$ and $b = \begin{pmatrix} \|\vec{w}_t\|^2 \\ y_t \end{pmatrix}$. Simple calculation shows that

$$
\begin{aligned}
\vec{w}_{t+1} &= A^T(AA^T)^{-1}b \\
&= \left( \frac{\|\vec{x}_t\|^2\|\vec{w}_t\|^2 - y_t(\vec{w}_t \cdot \vec{x}_t)}{\|\vec{x}_t\|^2\|\vec{w}_t\|^2 - (\vec{w}_t \cdot \vec{x}_t)^2} \right) \vec{w}_t + \left( \frac{\|\vec{w}_t\|^2(y_t - (\vec{w}_t \cdot \vec{x}_t))}{\|\vec{x}_t\|^2\|\vec{w}_t\|^2 - (\vec{w}_t \cdot \vec{x}_t)^2} \right) \vec{x}_t. \quad (5)
\end{aligned}
$$

If on trials $t$ in which a mistake is made, $c_t = \frac{\|\vec{x}_t\|^2\|\vec{w}_t\|^2 - y_t(\vec{w}_t \cdot \vec{x}_t)}{\|\vec{x}_t\|^2\|\vec{w}_t\|^2 - (\vec{w}_t \cdot \vec{x}_t)^2}$ and $d_t = \frac{\|\vec{w}_t\|^2(y_t - (\vec{w}_t \cdot \vec{x}_t))}{\|\vec{x}_t\|^2\|\vec{w}_t\|^2 - (\vec{w}_t \cdot \vec{x}_t)^2}$, and on other trials $c_t = 1$ and $d_t = 0$, then always $\vec{w}_{t+1} = c_t\vec{w}_t + d_t\vec{x}_t$. Note that based on Lemmas 1 and 2, the denominators in (5) will never be equal to zero.

Since the computations required by ROMMA involve inner products together with a few operations on scalars, we can apply the kernel method to our algorithm, efficiently solving the original problem in a very high dimensional space. Computationally, we only need to modify the algorithm by replacing each inner product computation $(\vec{x}_i \cdot \vec{x}_j)$ with a kernel function computation $\mathcal{K}(\vec{x}_i, \vec{x}_j)$.

To make a prediction for the $t$th trial, the algorithm must compute the inner product between $\vec{x}_t$ and prediction vector $\vec{w}_t$. In order to apply the kernel function, as in [1, 3], we store each prediction vector $\vec{w}_t$ in an implicit manner, as the weighted sum of examples on which

mistakes occur during the training. In particular, each $\vec{w}_t$ is represented as

$$\vec{w}_t = \left(\prod_{j=1}^{t-1} c_j\right)\vec{w}_1 + \sum_{j=1}^{t-1}\left(\prod_{n=j+1}^{t-1} c_n\right)d_j\vec{x}_j$$

The above formula may seem daunting; however, making use of the recurrence $(\vec{w}_{t+1}\cdot\vec{x}) = c_t(\vec{w}_t\cdot\vec{x}) + d_t(\vec{x}_t\cdot\vec{x})$, it is obvious that the complexity of our new algorithm is similar to that of perceptron algorithm. This was born out by our experiments.

The implementation for aggressive ROMMA is similar to the above.

## 4 Experiments

We did some experiments using the ROMMA and aggressive ROMMA as batch algorithms on the MNIST OCR database. [2] We obtained a batch algorithm from our online algorithm in the usual way, making a number of passes over the dataset and using the final weight vector to classify the test data.

Every example in this database has two parts, the first is a $28 \times 28$ matrix which represents the image of the corresponding digit. Each entry in the matrix takes value from $\{0,\cdots,255\}$. The second part is a label taking a value from $\{0,\cdots,9\}$. The dataset consists of $60,000$ training examples and $10,000$ test examples. We adopt the following polynomial kernel: $\mathcal{K}(\vec{x}_i,\vec{x}_j) = (1 + (\vec{x}_i\cdot\vec{x}_j))^d$. This corresponds to using an expanded collection of features including all products of at most $d$ components of the original feature vector (see [14]). Let us refer to the mapping from the original feature vector to the expanded feature vector as $\Phi$. Note that one component of $\Phi(\vec{x})$ is always 1, and therefore the component of the weight vector corresponding to that component can be viewed as a bias. In our experiments, we set $\vec{w}_1 = \Phi(\vec{0})$ rather than $\vec{0}$ to speed up the learning of the coefficient corresponding to the bias. We chose $d = 4$ since in experiments on the same problem conducted in [3, 2], the best results occur with this value.

To cope with multiclass data, we trained ROMMA and aggressive ROMMA once for each of the 10 labels. Classification of an unknown pattern is done according to the maximum output of these ten classifiers.

As every entry in the image matrix takes value from $\{0,\cdots,255\}$, the order of magnitude of $\mathcal{K}(\vec{x},\vec{x})$ is at least $10^{26}$, which might cause round-off error in the computation of $c_i$ and $d_i$. We scale the data by dividing each entry with 1100 when training with ROMMA.

Table 1: Experimental results on MNIST data

| | $T=1$ | | $T=2$ | | $T=3$ | | $T=4$ | |
|---|---|---|---|---|---|---|---|---|
| | Err | MisNo | Err | MisNo | Err | MisNo | Err | MisNo |
| percep | 2.84 | 7970 | 2.27 | 10539 | 1.99 | 11945 | 1.85 | 12800 |
| voted-percep | 2.26 | 7970 | 1.88 | 10539 | 1.76 | 11945 | 1.69 | 12800 |
| ROMMA | 2.48 | 7963 | 1.96 | 9995 | 1.79 | 10971 | 1.77 | 11547 |
| agg-ROMMA | 2.14 | 6077 | 1.82 | 7391 | 1.71 | 7901 | 1.67 | 8139 |
| agg-ROMMA(NC) | 2.05 | 5909 | 1.76 | 6979 | 1.67 | 7339 | 1.63 | 7484 |

Since the performance of online learning is affected by the order of sample sequence, all the results shown in Table 1 average over 10 random permutations. The columns marked

[2]National Institute for Standards and Technology, special database 3. See http://www.research.att.com/~yann/ocr for information on obtaining this dataset.

"MisNo" in Table 1 show the total number of mistakes made during the training for the 10 labels. Although online learning would involve only one epoch, we present results for a batch setting until four epochs ($T$ in Table 1 represents the number of epochs).

To deal with data which are linearly inseparable in the feature space, and also to improve generalization, Friess et al [4] suggested the use of quadratic penalty in the cost function, which can be implemented using a slightly different kernel function [4, 5]: $\tilde{\mathcal{K}}(x_k, x_j) = \mathcal{K}(x_k, x_j) + \delta_{kj}\lambda$, where $\delta_{kj}$ is the Kronecker delta function. The last row in Table 1 is the result of aggressive ROMMA using this method to control noise ($\lambda = 30$ for 10 classifiers).

We conducted three groups of experiments, one for the perceptron algorithm (denoted "percep"), the second for the voted perceptron (denoted "voted-percep") whose description is in [3], the third for ROMMA, aggressive ROMMA (denoted "agg-ROMMA"), and aggressive ROMMA with noise control (denoted "agg-ROMMA(NC)"). Data in the third group are scaled. All three groups set $\vec{w}_1 = \Phi(\vec{0})$.

The results in Table 1 demonstrate that ROMMA has better performance than the standard perceptron, aggressive ROMMA has slightly better performance than the voted perceptron. Aggressive ROMMA with noise control should not be compared with perceptrons without noise control. Its presentation is used to show what performance our new online algorithm could achieve (of course it's not the best, since all 10 classifiers use the same $\lambda$). A remarkable phenomenon is that our new algorithm behaves very well at the first two epochs.

## Footnotes

[1]The prediction of 0, which ensures a mistake, is to make the proofs simpler. The usual mistake bound proof for the perceptron algorithm goes through with this change.

# References

[1] B. E. Boser, I. M. Guyon, and V. N. Vapnik. A training algorithm for optimal margin classifiers. *Proceedings of the 1992 Workshop on Computational Learning Theory*, pages 144–152, 1992.

[2] C. Cortes and V. Vapnik. Support-vector networks. *Machine Learning*, 20(3):273–297, 1995.

[3] Y. Freund and R. E. Schapire. Large margin classification using the perceptron algorithm. *Proceedings of the 1998 Conference on Computational Learning Theory*, 1998.

[4] T. T. Friess, N. Cristianini, and C. Campbell. The kernel adatron algorithm: a fast and simple learning procedure for support vector machines. In *Proc. 15th Int. Conf. on Machine Learning*. Morgan Kaufman Publishers, 1998.

[5] S. S. Keerthi, S. K. Shevade, C. Bhattacharyya, and K. R. K. Murthy. A fast iterative nearest point algorithm for support vector machine classiifer design. Technical report, Indian Institute of Science, 99. TR-ISL-99-03.

[6] Adam Kowalczyk. Maximal margin perceptron. In Smola, Bartlett, Scholkopf, and Schuurmans, editors, *Advances in Large Margin Classifiers*, 1999. MIT-Press.

[7] N. Littlestone. Learning quickly when irrelevant attributes abound: a new linear-threshold algorithm. *Machine Learning*, 2:285–318, 1988.

[8] N. Littlestone. *Mistake Bounds and Logarithmic Linear-threshold Learning Algorithms*. PhD thesis, UC Santa Cruz, 1989.

[9] John C. Platt. Fast training of support vector machines using sequential minimal optimization. In B. Scholkopf, C. Burges, A. Smola, editors, *Advances in Kernel Methods: Support Vector Machines*, 1998. MIT Press.

[10] F. Rosenblatt. The perceptron: A probabilistic model for information storage and organization in the brain. *Psychological Review*, 65:386–407, 1958.

[11] F. Rosenblatt. *Principles of Neurodynamics: Perceptrons and the Theory of Brain Mechanisms*. Spartan Books, Washington, D. C., 1962.

[12] J. Shawe-Taylor, P. Bartlett, R. Williamson, and M. Anthony. A framework for structural risk minimization. In *Proc. of the 1996 Conference on Computational Learning Theory*, 1996.

[13] V. N. Vapnik. *Estimation of Dependencies based on Empirical Data*. Springer Verlag, 1982.

[14] V. N. Vapnik. *The Nature of Statistical Learning Theory*. Springer, 1995.
